# Neural Networks Structured for Control Application to Aircraft Landing

**Charles Schley, Yves Chauvin, Van Henkle, Richard Golden**
Thomson-CSF, Inc., Palo Alto Research Operations
630 Hansen Way, Suite 250
Palo Alto, CA 94306

## Abstract

We present a generic neural network architecture capable of controlling non-linear plants. The network is composed of dynamic, parallel, linear maps gated by non-linear switches. Using a recurrent form of the back-propagation algorithm, control is achieved by optimizing the control gains and task-adapted switch parameters. A mean quadratic cost function computed across a nominal plant trajectory is minimized along with performance constraint penalties. The approach is demonstrated for a control task consisting of landing a commercial aircraft in difficult wind conditions. We show that the network yields excellent performance while remaining within acceptable damping response constraints.

## 1 INTRODUCTION

This paper illustrates how a recurrent back-propagation neural network algorithm (Rumelhart, Hinton & Williams, 1986) may be exploited as a procedure for controlling complex systems. In particular, a simplified mathematical model of an aircraft landing in the presence of severe wind gusts was developed and simulated. A recurrent back-propagation neural network architecture was then designed to numerically estimate the parameters of an optimal non-linear control law for landing the aircraft. The performance of the network was then evaluated.

### 1.1 A TYPICAL CONTROL SYSTEM

A typical control system consists of a controller and a process to be controlled. The controller's function is to accept task inputs along with process outputs and to determine control signals tailored to the response characteristics of the process. The

physical process to be controlled can be electro–mechanical, aerodynamic, etc. and generally has well defined behavior. It may be subjected to disturbances from its external environment.

## 1.2 CONTROLLER DESIGN

Many variations of both classical and modern methods to design control systems are described in the literature. Classical methods use linear approximations of the plant to be controlled and some loosely defined response specifications such as bandwidth (speed of response) and phase margin (degree of stability). Classical methods are widely used in practice, even for sophisticated control problems. Modern methods are more universal and generally assume that a performance index for the process is specified. Controllers are then designed to optimize the performance index. Our approach relates more to modern methods.

Narendra and Parthasarathy (1990) and others have noted that recurrent back–propagation networks can implement gradient descent algorithms that may be used to optimize the performance of a system. The essence of such methods is first to propagate performance errors back through the process and then back through the controller to give error signals for updating the controller parameters. Figure 1 provides an overview of the interaction of a neural control law with a complex system and possible performance indices for evaluating various control laws. The functional components needed to train the controller are shown within the shaded box of Figure 1. The objective performance measure contains factors that are written mathematically and usually represent terms such as weighted square error or other quantifiable measures. The performance constraints are often more subjective in nature and can be formulated as

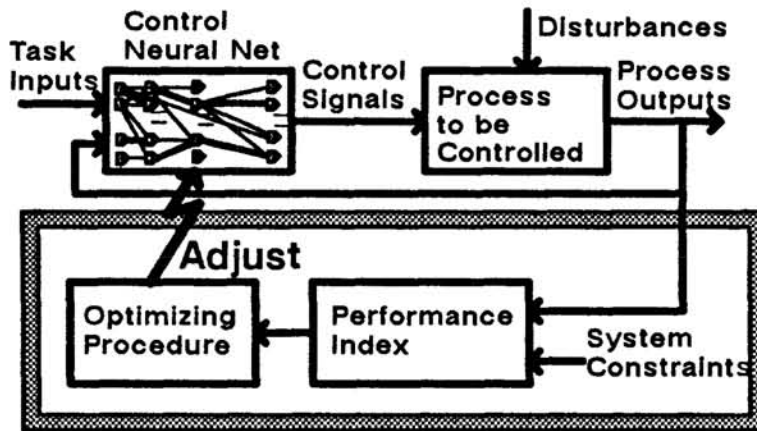

Figure 1: Neural Network Controller Design

reward or penalty functions on categories of performance (e.g., "good" or "bad").

# 2 A GENERIC NON–LINEAR CONTROL ARCHITECTURE

Many complex systems are in fact non–linear or "multi–modal." That is, their behavior changes in fundamental ways as a function of their position in the state space. In practice, controllers are often designed for such systems by treating them as a collection of systems linearized about a "setpoint" in state space. A linear controller can then be determined separately for each of these system "modes." These observations suggest that a reasonable approach for controlling non–linear or "multi–modal" systems would be to design a "multi–modal" control law.

## 2.1 THE SWITCHING PRINCIPLE

The architecture of our proposed general non–linear control law for "multi–mod-al" plants is shown in Figure 2. Task inputs and process outputs are entered into

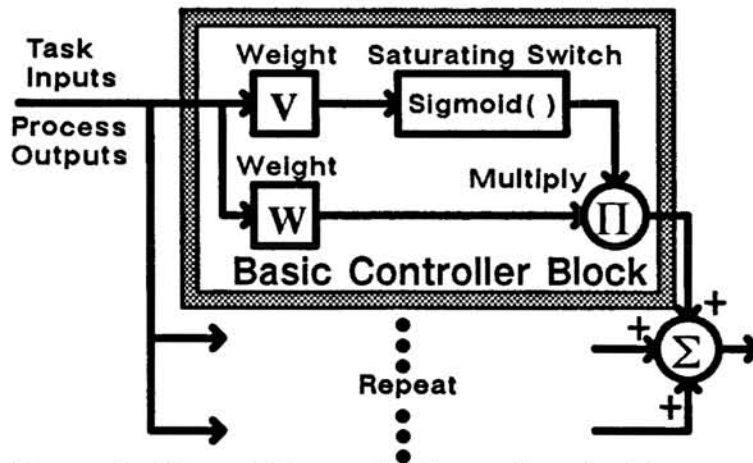

Figure 2: Neural Network Controller Architecture

multiple basic controller blocks (shown within the shaded box of Figure 2). Each basic controller block first determines a weighted sum of the task inputs and process outputs (multiplication by weights $W$). Then, the degree to which the weighted sum passes through the block is modified by means of a saturating switch and multiplier. The input to the switch is itself another weighted sum of the task inputs and process outputs (multiplication by weights $V$). If the input to the saturating switch is large, its output is unity and the weighted sum (weighted by $W$) is passed through unchanged. At the other extreme, if the saturating switch has zero output, the weighted sum of task inputs and process outputs does not appear in the output. When these basic controller blocks are replicated and their outputs are added, control signals consist of weighted sums of the controller inputs that can be selected and/or blended by the saturating switches. The overall effect is a prototypical feed–forward and feedback controller with selectable gains and multiple pathways where the overall equivalent gains are a function of the task inputs and process outputs. The resulting architecture yields a *sigma–pi* processing unit in the final controller (Rumelhart, Hinton & Williams, 1986).

## 2.2 MODELLING DYNAMIC MAPPINGS

Weights shown in Figure 2 may be constant and represent a static relationship between input and control. However, further controller functionality is obtained by considering the weights $V$ and $W$ as dynamic mappings. For example, proportional plus integral plus derivative (PID) feedback may be used to ensure that process outputs follow task inputs with adequate steady–state error and transient damping. Thus, the weights can express parameters of various generally useful control functions. These functions, when combined with the switching principle, yield rich capabilities that can be adapted to the task at hand.

## 3 AIRCRAFT LANDING

The generic neural network architecture of Figure 2 and the associated neural network techniques were tested with a "real–world" application: automatic landing of an aircraft. Here, we describe the aircraft and environment model during landing.

### 3.1 GLIDESLOPE AND FLARE

During aircraft landing, the final two phases of a landing trajectory consist of a "glideslope" phase and a "flare" phase. Figure 3 shows these two phases. Flare occurs at about 45 feet. Glideslope is characterized by a linear downward slope; flare by an exponential shaped curve. When the aircraft begins flare, its response characteristics are changed to make it more sensitive to the pilot's actions, making the process "multi–modal" or non–linear over the whole trajectory.

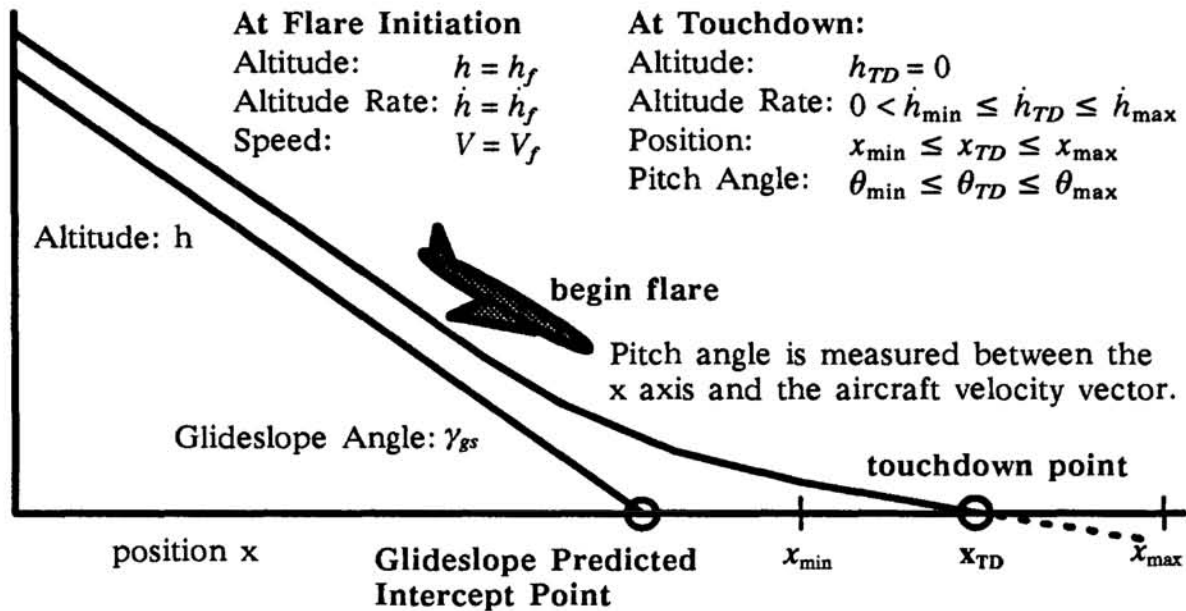

**At Flare Initiation**
Altitude:       $h = h_f$
Altitude Rate: $\dot{h} = \dot{h}_f$
Speed:           $V = V_f$

**At Touchdown:**
Altitude:        $h_{TD} = 0$
Altitude Rate: $0 < \dot{h}_{min} \le \dot{h}_{TD} \le \dot{h}_{max}$
Position:       $x_{min} \le x_{TD} \le x_{max}$
Pitch Angle:   $\theta_{min} \le \theta_{TD} \le \theta_{max}$

Altitude: h

begin flare

Pitch angle is measured between the x axis and the aircraft velocity vector.

Glideslope Angle: $\gamma_{gs}$

touchdown point

position x

Glideslope Predicted Intercept Point

$x_{min}$     $x_{TD}$     $x_{max}$

Figure 3: Glideslope and Flare Geometry

## 3.2 STATE EQUATIONS

Linearized equations of motion were used for the aircraft during each phase. They are adequate during the short period of time spent during glideslope and flare. A pitch stability augmentation system and an auto–throttle were added to the aircraft state equations to damp the bare airframe oscillatory behavior and provide speed control. The function of the autoland controller is to transform information about desired and actual trajectories into the aircraft pitch command. This is input to the pitch stability augmentation system to develop the aircraft elevator angle that in turn controls the aircraft's actual pitch angle. Simplifications retain the overall quality of system response (i.e., high frequency dynamics were neglected).

## 3.3 WIND MODEL

The environment influences the process through wind disturbances represented by constant velocity and turbulence components. The magnitude of the constant velocity component is a function of altitude (wind shear). Turbulence is a stochastic process whose mean and variance are functions of altitude. For the horizontal and vertical wind turbulence velocities, the so–called Dryden spectra for spatial turbulence distribution are assumed. These are amenable to simulation and show reasonable agreement with measured data (Neuman & Foster, 1970). The generation of turbulence is effected by applying Gaussian white noise to coloring filters.

## 4  NEURAL NETWORK LEARNING IMPLEMENTATION

As previously noted, modern control theory suggests that a performance index for evaluating control laws should first be constructed, and then the control law should be computed to optimize the performance index. Generally, numerical methods are used for estimating the parameters of a control law. Neural network algorithms can actually be seen as constituting such numerical methods (Narendra and Parthasarathy, 1990; Bryson and Ho, 1969; Le Cun, 1989). We present here an implementation of a neural network algorithm to address the aircraft landing problem.

## 4.1 DIFFERENCE EQUATIONS

The state of the aircraft (including stability augmentation and autothrottle) can be represented by a vector $X$, containing variables representing speed, angle of attack, pitch rate, pitch angle, altitude rate and altitude. The difference equations describing the dynamics of the controlled plant can be written as shown in equation 1.

$$X_{t+1} = A_t X_t + B_t U_t + C D_t + N_t \qquad (1)$$

The matrix $A$ represents the plant dynamics and $B$ represents the aircraft response to the control $U$. $D$ is the desired state and $N$ is the additive noise computed from the wind model. The switching controller can be written as in equation 2 below. Referring to Figure 2, the weight matrix $V$ in the sigmoidal switch links actual altitude to each switch unit. The weight matrix $W$ in the linear controller links altitude error, altitude rate error and altitude integral error to each linear unit output.

$$U_t = P_t^T L_t \quad \text{where } P_t = \text{ Sigmoidal switch and } L_t = \text{ Linear controller} \qquad (2)$$

Figure 4 shows a recurrent network implementation of the entire system. Actual and desired states at time $t+1$ are fed back to the input layers. Thus, with recurrent connections between output and input, the network generates entire trajectories and is seen as a recurrent back–propagation network (Rumelhart, Hinton & Williams, 1986; Jordan & Jacobs, 1990). The network is trained using the back–propagation algorithm with given wind distributions. For the controller, we chose initially two basic PID controller blocks (see Figure 2) to represent glideslope and flare. The task of the network is then to learn the state dependent PID controller gains that optimize the cost function.

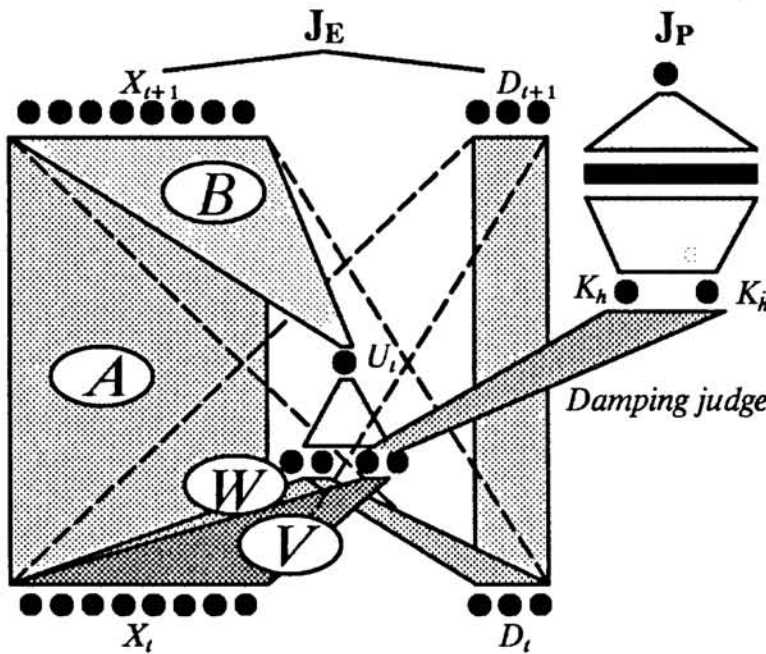

Figure 4: Recurrent Neural Network Architecture

## 4.2 PERFORMANCE INDEX OPTIMIZATION

The basic performance measure selected for this problem was the squared trajectory error accumulated over the duration of the landing. Trajectory error corresponds to a weighted combination of altitude and altitude rate errors.

Since minimizing only the trajectory error can lead to undesirable responses (e.g., oscillatory aircraft motions), we include relative stability constraints in the performance index. Aircraft transient responses depend on the value of a damping factor. An analysis was performed by probing the plant with a range of values for controller weight parameters. The data was used to train a "damping judge" net to categorize "good" and "bad" damping. This net was used to construct a penalty function on "bad" damping. As seen in Figure 4, additional units were added for this purpose.

The main optimization problem is now stated. Given an initial state, minimize the expected value over the environmental disturbances of performance index $J$.

$$J = J_E + J_{P} = \text{Trajectory Error } + \text{ Performance Constraint Penalty} \tag{3}$$

$$J_E = \sum_{t=1}^{T} a_h[ h_{cmd_t} - h_t ]^2 + a_{\dot{h}}[ \dot{h}_{cmd_t} - \dot{h}_t ]^2 \quad \text{We used } a_h = a_{\dot{h}} = 1$$

$$J_P = \sum_{t=1}^{T} Max(0, \zeta_{judge} - \zeta^{\bullet}_{judge})(\zeta_{judge} - \zeta^{\bullet}_{judge}) \quad \begin{array}{l} \textit{Note that when } \zeta_{judge} \leq \zeta^{\bullet}_{judge}, \textit{ there is} \\ \textit{no penalty. Otherwise, it is quadratic.} \end{array}$$

## 5 SIMULATION EXPERIMENTS

We now describe our simulations. First, we introduce our training procedure. We then present statistical results of flight simulations for a variety of wind conditions.

### 5.1 TRAINING PROCEDURE

Networks were initialized in various ways and trained with a random wind distribution where the constant sheared speed varied from 10 ft/sec tailwind to 40 ft/sec headwind (a strong wind). Several learning strategies were used to change the way the network was exposed to various response characteristics of the plant. The exact form of the resulting $V$ switch weights varied, but not the equivalent gain schedules.

### 5.2 STATISTICAL RESULTS

After training, the performance of the network controller was tested for different wind conditions. Table 1 shows means and standard deviations of performance variables computed over 1000 landings for five different wind conditions. Shown

Table 1: Landing Statistics (standard deviations in parenthesis).

| | Overall Performance | | Glide Slope Mean Squared Error | | Flare Mean Squared Error | | Touchdown Performance | | |
|---|---|---|---|---|---|---|---|---|---|
| Wind | $J/T$ | $T$ | $J : h_{gs}$ | $J : \dot{h}_{gs}$ | $J : h_{fl}$ | $J : \dot{h}_{fl}$ | $x_{TD}$ | $\theta_{TD}$ | $h_{TD}$ |
| H=-10 | 1.34 | 21.6 | 13.500 | 4.230 | 3.50 | 2.67 | 1030 | 0.0473 | -2.15 |
| | (0 56) | (0.047) | (9.8) | (2.6) | (4.4) | (1.2) | (11) | (0.0039) | (0.052) |
| H=0 | 0.27 | 22.2 | 0.603 | 0.209 | 3.31 | 1 86 | 1080 | -0.0400 | -1.98 |
| | (0.00) | (0.000) | (0.0) | (0.0) | (0.0) | (0 0) | (0) | (0.0000) | (0.000) |
| H=10 | 0.96 | 23.0 | 12.500 | 4.390 | 3.17 | 2.01 | 1160 | -0.1260 | -1.79 |
| | (0.53) | (0.052) | (9.2) | (2.6) | (2.2) | (1.1) | (12) | (0.0046) | (0.040) |
| H=20 | 3.56 | 23.8 | 54.000 | 18.500 | 3.65 | 3.43 | 1230 | -0.2170 | -1.64 |
| | (2.10) | (0.100) | (38.0) | (11.0) | (8.2) | (2.7) | (24) | (0.0100) | (0.061) |
| H=30 | 8.03 | 24.6 | 130.000 | 43.200 | 19.20 | 5.73 | 1310 | -0.3110 | -1.50 |
| | (4 70) | (0.160) | (91.0) | (25.0) | (19.0) | (4 7) | (39) | (0.0170) | (0.076) |
| H=40 | 13.40 | 25.5 | 219 000 | 76 200 | 37 00 | 9 24 | 1400 | -0.4030 | -1.39 |
| | (7 80) | (0.220) | (150.0) | (46.0) | (36.0) | (8.0) | (54) | (0.0230) | (0.083) |

are values for overall performance (quadratic cost $J$ per time step, landing time $T$), trajectory performance (quadratic cost $J$ on altitude and altitude rate), and landing performance (touchdown position, pitch angle, altitude rate).

## 5.3 CONTROL LAWS OBTAINED BY LEARNING

By examining network weights, equation 2 yields the gains of an equivalent controller over the entire trajectory (gain schedules). These gain schedules represent *optimality* with respect to a given performance index. Results show that the switch builds a *smooth* transition between glideslope and flare and provides the network controller with a *non-linear distributed* control law for the whole trajectory.

# 6 DISCUSSION

The architecture we propose integrates *a priori* knowledge of real plants within the structure of the neural network. The knowledge of the physics of the system and its representation in the network are part of the solution. Such *a priori* knowledge structures are not only useful for finding control solutions, but also allow interpretations of network dynamics in term of standard control theory. By observing the weights learned by the network, we can compute gain schedules and understand how the network controls the plant.

The augmented architecture also allows us to control damping. In general, integrating optimal control performance indices with constraints on plant response characteristics is not an easy task. The neural network approach and back–propagation learning represent an interesting and elegant solution to this problem. Other constraints on states or response characteristics can also be implemented with similar architectures. In the present case, the control gains are obtained to minimize the objective performance index while the plant remains within a desired stability region. The effect of this approach provides good damping and control gain schedules that make the plant robust to disturbances.

### Acknowledgements

This research was supported by the Boeing High Technology Center. Particular thanks are extended to Gerald Cohen of Boeing. We would also like to thank Anil Phatak for his decisive help and Yoshiro Miyata for the use of his XNet simulator.

### References

Bryson, A. & Ho, Y. C. (1969). *Applied Optimal Control.* Blaisdel Publishing Co.

Jordan, M. I. & Jacobs, R. A. (1990). Learning to control an unstable system with forward modeling. In D. S. Touretzky (Ed.), *Neural Information Processing Systems 2.* Morgan Kaufman: San Mateo, CA.

Le Cun, Y. (1989). A theoretical framework for back–propagation. In D. Touretzky, G. Hinton and T. Sejnowski (Eds.), *Proceedings of the 1988 Connectionist Models Summer School.* Morgan Kaufman: San Mateo, CA.

Narendra, K. & Parthasarathy, K. (1990). Identification and control of dynamical systems using neural networks. *IEEE Transactions on Neural Networks, 1,* 4–26.

Neuman, F. & Foster, J. D. (1970). Investigation of a digital automatic aircraft landing system in turbulence. *NASA Technical Note TN D-6066.* NASA–Ames Research Center, Moffett Field, CA.

Rumelhart, D. E., Hinton G. E., Williams R. J. (1986). Learning internal representations by error propagation. In D. E. Rumelhart & J. L. McClelland (Eds.) *Parallel Distributed Processing: Explorations in the Microstructures of Cognition (Vol. I).* Cambridge, MA: MIT Press.